# A Computational Geometric Approach to Shape Analysis in Images

**Anuj Srivastava**
Department of Statistics
Florida State University
Tallahassee, FL 32306
anuj@stat.fsu.edu

**Washington Mio**
Department of Mathematics
Florida State University
Tallahassee, FL 32306
mio@math.fsu.edu

**Xiuwen Liu**
Department of Computer Science
Florida State University
Tallahassee, FL 32306
liux@cs.fsu.edu

**Eric Klassen**
Department of Mathematics
Florida State University
Tallahassee, FL 32306
klassen@math.fsu.edu

## Abstract

We present a geometric approach to statistical shape analysis of closed curves in images. The basic idea is to specify a space of closed curves satisfying given constraints, and exploit the differential geometry of this space to solve optimization and inference problems. We demonstrate this approach by: (i) defining and computing statistics of observed shapes, (ii) defining and learning a parametric probability model on shape space, and (iii) designing a binary hypothesis test on this space.

## 1   Introduction

An important goal in image understanding is to detect, track and label objects of interest present in observed images. Imaged objects can be characterized in many ways: according to their colors, textures, shapes, movements, and locations. The past decade has seen significant advances in the modeling and analysis of pixel values or textures to characterize objects in images, albeit with limited success. On the other hand, planar curves that represent contours of objects have been studied independently for a long time. An emerging opinion in the vision community is that global features such as shapes of contours should also be taken into account for the successful detection and recognition of objects. A common approach to analyzing curves in images is to treat them as level sets of functions, and algorithms involving such *active contours* are governed usually by partial differential equations (PDEs) driven by appropriate data terms and smoothness penalties (see for example [10]). Regularized curve evolutions and region-based active contours offer alternatives in similar frameworks. This remarkable body of work contains various studies of curve evolution, each with relative strengths and drawbacks.

In this paper, we present a framework for the algorithmic study of curves, their variations and statistics. In this approach, a fundamental element is a space of closed curves,

with additional constraints to impose equivalence of shapes under rotation, translation, and scale. We exploit the geometry of these spaces using elements such as tangents, normals, geodesics and gradient flows, to solve optimization and statistical inference problems for a variety of cost functions and probability densities. This framework differs from those employed in previous works on "geometry-driven flows" [8] in the sense that here both the geometry of the curves and the geometry of spaces of curves are utilized. Here the dynamics of active contours is described by vector fields on spaces of curves. It is important to emphasize that a shape space is usually a non-linear, infinite-dimensional manifold, and its elements are the individual curves of interest. Several interesting applications can be addressed in this formulation, including: 1) Efficient deformations between any two curves are generated by geodesic paths connecting the elements they represent in the shape space. Geodesic lengths also provide a natural metric for shape comparisons. 2) Given a set of curves (or shapes), one can define the concepts of *mean* and *covariance* using geodesic paths, and thus develop statistical frameworks for studying shapes. Furthermore, one can define probabilities on a shape space to perform curve (or shape) classification via hypothesis testing. While these problems have been studied in the past with elegant solutions presented in the literature (examples include [9, 11, 7, 2, 5]), we demonstrate the strength of the proposed framework by addressing these problems using significantly different ideas.

*Given past achievements in PDE-based approaches to curve evolution, what is the need for newer frameworks?* The study of the structure of the shape space provides new insights and solutions to problems involving dynamic contours and problems in quantitative shape analysis. Once the constraints are imposed in definitions of shape spaces, the resulting solutions automatically satisfy these constraints. It also complements existing methods of image processing and analysis well by realizing new computational efficiencies. The main strength of this approach is its exploitation of the differential geometry of the shape space. For instance, a geodesic or gradient flow $X_t$ of an energy function $E$ can be generated as a solution of an ordinary differential equation of the type

$$\frac{dX_t}{dt} = \Pi(\nabla E(X_t)) \,, \tag{1}$$

where $\Pi$ denotes an appropriate projection onto a tangent space. This contrasts with the nonlinear PDE-based curve evolutions of past works. The geometry of shape space also enables us to derive statistical elements: probability measures, means and covariances; these quantities have rarely been treated in previous studies. In shape extraction, the main focus in past works has been on solving PDEs driven by image features under smoothness constraints, and not on the statistical analysis of shapes of curves. The use of geodesic paths, or piecewise geodesic paths, has also seen limited use in the past.

We should also point out the main limitations of the proposed framework. One drawback is that curve evolutions can not handle certain changes in topology, which is one of the key features of level-set methods; a shape space is purposely setup to not allow curves to branch into several components. Secondly, this idea does not extend easily to the analysis of surfaces in $\mathbb{R}^3$. Despite these limitations, the proposed methodology provides powerful algorithms for the analysis of planar curves as demonstrated by the examples presented later. Moreover, even in applications where branching appears to be essential, the proposed methods may be applicable with additional developments.

This paper is laid out as follows: Section 2 studies geometric representations of constrained curves as elements of a shape space. Geometric analysis tools on the shape space are presented in Section 3. Section 4 provides examples of statistical analysis on the shape space, while Section 5 concludes the paper with a brief summary.

## 2 Representations of Shapes

In this paper we restrict the discussion to curves in $\mathbb{R}^2$ although curves in $\mathbb{R}^3$ can be handled similarly. Let $\alpha : \mathbb{R} \mapsto \mathbb{R}^2$ denote the coordinate function of a curve parametrized by arclength, i.e., satisfying $\|\dot{\alpha}(s)\| = 1$, for every $s$. A direction function $\theta(s)$ is a function satisfying $\dot{\alpha}(s) = e^{j\,\theta(s)}$, where $j = \sqrt{-1}$. $\theta$ captures the angle made by the velocity vector with the $x$-axis, and is defined up to the addition of integer multiples of $2\pi$. The curvature function $\kappa(s) = \dot{\theta}(s)$ can also be used to represent a curve.

Consider the problem of studying shapes of contours or silhouettes of imaged objects as closed, planar curves in $\mathbb{R}^2$, parametrized by arc length. Since shapes are invariant to rigid motions (rotations and translations) and uniform scaling, a shape representation should be insensitive to these transformations. Scaling can be resolved by fixing the length of $\alpha$ to be $2\pi$, and translations by representing curves via their direction functions. Thus, we consider the space $\mathbb{L}^2$ of all square integrable functions $\theta \colon [0, 2\pi] \to \mathbb{R}$, with the usual inner product $\langle f, g \rangle = \int_0^{2\pi} f(s)g(s)\,ds$. To account for rotations and ambiguities on the choice of $\theta$, we restrict direction functions to those having a fixed average, say, $\pi$. For $\alpha$ to be closed, it must satisfy the *closure condition* $\int_0^{2\pi} e^{j\theta(s)}\,ds = 0$. Thus, we represent curves by direction functions satisfying the average-$\pi$ and closure conditions; we call this space of direction functions $\mathcal{D}$. Summarizing, $\mathcal{D}$ is the subspace of $\mathbb{L}^2$ consisting of all (direction) functions satisfying the constraints

$$\frac{1}{2\pi} \int_0^{2\pi} \theta(s)\,ds \; = \pi \,; \quad \int_0^{2\pi} \cos(\theta(s))\,ds = \; 0 \,; \quad \int_0^{2\pi} \sin(\theta(s))\,ds = \; 0 \,. \qquad (2)$$

It is still possible to have multiple continuous functions in $\mathcal{D}$ representing the same shape. This variability is due to the choice of the reference point $(s = 0)$ along the curve. For $x \in \mathbb{S}^1$ and $\theta \in \mathcal{D}$, define $(x \cdot \theta)$ as a curve whose initial point $(s = 0)$ is changed by a distance of $x$ along the curve. We term this a re-parametrization of the curve. To remove the variability due to this re-parametrization group, define the quotient space $\mathcal{C} \equiv \mathcal{D}/\mathbb{S}^1$ as the space of continuous, planar shapes. For details, please refer to the paper [4].

## 3 Geometric Tools for Shape Analysis

The main idea in the proposed framework is to use the geometric structure of a shape space to solve optimization and statistical inference problems on these spaces. This approach often leads to simple formulations of these problems and to more efficient vision algorithms. Thus, we must study issues related to the differential geometry and topology of a shape space. In this paper we restrict to the tangent and normal bundles, exponential maps, and their inverses on these spaces.

### 3.1 Tangents and Normals to Shape Space

The main reason for studying the tangential and normal structures is the following: We wish to employ iterative numerical methods in the simulation of geodesic and gradient flows on the shape space. At each step in the iteration, we first flow in the linear space $\mathbb{L}^2$ using standard methods, and then project the new point back onto the shape space using our knowledge of the normal structure.

For technical reasons, it is convenient to reduce optimization and inference problems on $\mathcal{C}$ to problems on the manifold $\mathcal{D}$, so we study the latter. It is difficult to specify the tangent spaces to $\mathcal{D}$ directly, because they are infinite-dimensional. When working with finitely many constraints, as is the case here, it is easier to describe the space of normals to $\mathcal{D}$ in

$\mathbb{L}^2$ instead. It can be shown that a vector $f \in \mathbb{L}^2$ is tangent to $\mathcal{D}$ at $\theta$ if and only if $f$ is orthogonal to the subspace spanned by $\{1, \sin \theta, \cos \theta\}$. Hence, these three functions span the normal space to $\mathcal{D}$ at $\theta$. Implicitly, the tangent space is given as: $T_\theta(\mathcal{D}) = \{f \in \mathbb{L}^2 | f \perp \mathrm{span}\{1, \cos \theta, \sin \theta\}\}$. Thus, the projection $\Pi$ in Eqn. 1 can be specified by subtracting from a function (in $\mathbb{L}^2$) its projection onto the space spanned by these three elements.

## 3.2 Exponential Maps

We first describe the computation of geodesics (or, one-parameter flows) in $\mathcal{D}$ with pre-scribed initial conditions. Geodesics on $\mathcal{D}$ are realized as exponential maps from tangent spaces to $\mathcal{D}$. The intricate geometry of $\mathcal{D}$ disallows explicit analytic expressions. There-fore, we adopt an iterative strategy, where in each step, we first flow infinitesimally in the prescribed tangent direction in the space $\mathbb{L}^2$, and then project the end point of the path to $\mathcal{D}$. Next, we parallel transport the velocity vector to the new point by projecting the previous velocity orthogonally onto the tangent space of $\mathcal{D}$ at the new point. Again, this is done by subtracting normal components. The simplest implementation is to use Euler's method in $\mathbb{L}^2$, i.e., to move in each step along short straight line segments in $\mathbb{L}^2$ in the prescribed direction, and then project the path back onto $\mathcal{D}$. Details of this numerical construction of geodesics are provided in [4].

A geodesic can be specified by an initial condition $\theta \in \mathcal{D}$ and a direction $f \in T_\theta(\mathcal{D})$, the space of all tangent directions at $\theta$. We will denote the corresponding geodesic by $\Psi(\theta, t, f)$, where $t$ is the time parameter. The technique just described allows us to compute $\Psi$ numerically. For $t = 1$, $\Psi(\theta, 1, f)$ is the exponential map from $f \in T_\theta \mathcal{D}$ to $\mathcal{D}$.

## 3.3 Shape Logarithms

Next, we focus on the problem of finding a geodesic path between any two given shapes $\theta_1, \theta_2 \in \mathcal{D}$. This is akin to inverting the exponential map. The main issue is to find that appropriate direction $f \in T_{\theta_1}(\mathcal{D})$ such that a geodesic from $\theta_1$ in that direction passes through $\theta_2$ at time $t = 1$. In other words, the problem is to solve for an $f \in T_{\theta_1}(\mathcal{D})$ such that $\Psi(\theta_1, 0, f) = \theta_1$ and $\Psi(\theta_1, 1, f) = \theta_2$. One can treat the search for this direction as an optimization problem over the tangent space $T_{\theta_1}(\mathcal{D})$. The cost to be minimized is given by the functional $H[f] = \|\Psi(\theta_1, 1, f) - \theta_2\|^2$, and we are looking for that $f \in T_{\theta_1}(\mathcal{C})$ for which: (i) $H[f]$ is zero, and (ii) $\|f\|$ is minimum among all such tangents. Since the space $T_{\theta_1}(\mathcal{D})$ is infinite dimensional, this optimization is not straightforward. However, since $f \in \mathbb{L}^2$, it has a Fourier decomposition, and we can solve the optimization problem over a finite number of Fourier coefficients. For any two shapes $\theta_1, \theta_2 \in \mathcal{D}$, we have used a shooting method to find the optimal $f$ [4]. The basic idea is to choose an initial direction $f$ specified by its Fourier coefficients and then use a gradient search to minimize $H$ as a function of the Fourier coefficients.

Finally, to find the shortest path between two shapes in $\mathcal{C}$, we compute the shortest geodesic connecting representatives of the given shapes in $\mathcal{D}$. This is a simple numerical problem, because $\mathcal{C}$ is the quotient of $\mathcal{D}$ by the 1-dimensional re-parametrization group $\mathbb{S}^1$. Shown in Figure 1 are three examples of geodesic paths in $\mathcal{C}$ connecting given shapes. Drawn in between are shapes corresponding to equally spaced points along the geodesic paths.

# 4 Statistical Analysis on Shape Spaces

Our goal is to develop tools for statistical analysis of shapes. Towards that goal, we develop the following ideas.

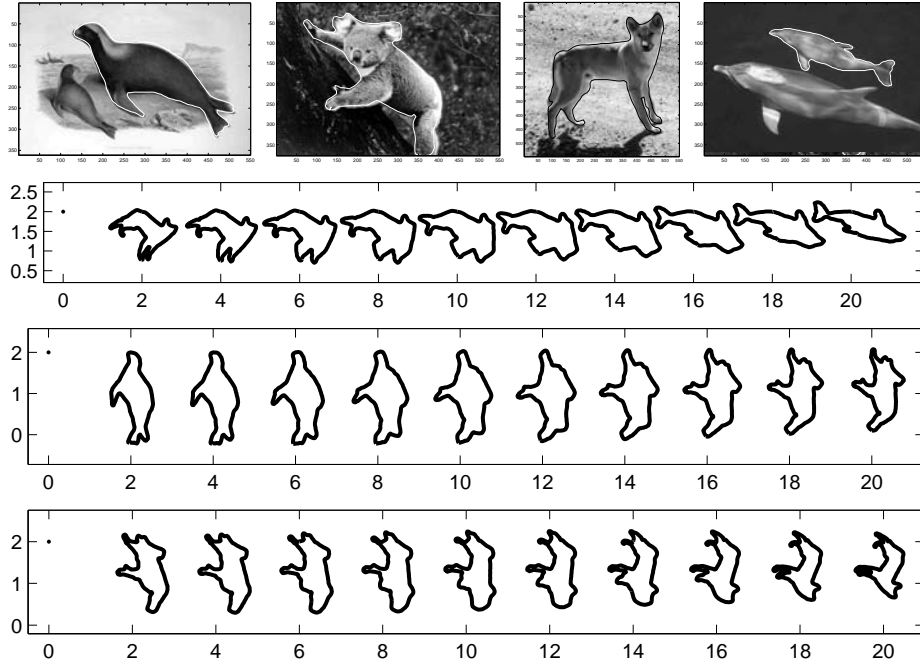

Figure 1: Top panels show examples of shapes manually extracted from the images. Bottom panels show examples of evolving one shape into another via a geodesic path. In each case, the leftmost shape is $\theta_1$, rightmost curves are $\theta_2$, and intermediate shapes are equi-spaced points along the geodesic.

## 4.1 Sample Means on Shape Spaces

Algorithms for finding geodesic paths on the shape space allow us to compute means and covariances in these spaces. We adopt a notion of mean known as the *intrinsic mean* or the *Karcher mean* ([3]) that is quite natural in our geometric framework. Let $d(\_,\_)$ be the shortest-path metric on $\mathcal{C}$. To calculate the Karcher mean of shapes $\{\theta_1,\ldots,\theta_n\}$ in $\mathcal{C}$, define a function $V : \mathcal{C} \to \mathbb{R}$ by $V(\theta) = \sum_{i=1}^{n} d(\theta,\theta_i)^2$. Then, define the *Karcher mean* of the given shapes to be any point $\mu \in \mathcal{C}$ for which $V(\mu)$ is a local minimum. In the case of Euclidean spaces this definition agrees with the usual definition $\mu = \frac{1}{n} \sum_{i=1}^{n} p_i$. Since $\mathcal{C}$ is complete, the intrinsic mean as defined above always exists. However, there may be collections of shapes for which $\mu$ is not unique. An iterative algorithm for finding a Karcher mean of given shapes is given in [4] and see [4] for details.

## 4.2 Shape Learning

Another important problem in statistical analysis of shapes is to "learn" probability models from the observed shapes. Once the shapes are clustered, we assume that elements in the same cluster are (random) samples from the same probability model, and try to learn this model. These models can then be used for future Bayesian discoveries of shapes or for classification of new shapes. To learn a probability model amounts to estimating a probability density function on the shape space, a task that is rather difficult to perform precisely. The two main difficulties are: nonlinearity and infinite-dimensionality of $\mathcal{C}$, and they are handled here as follows.

1. **Tangent Space**: Since $\mathcal{C}$ is a nonlinear manifold, we impose a probability density on a tangent space rather than on $\mathcal{C}$ directly. For a mean shape $\mu \in \mathcal{C}$, the space of all tangents to the shape space at $\mu$, $T_\mu(\mathcal{C}) \subset \mathbb{L}^2$, is an infinite-dimensional **vector space**. Similar to the ideas presented in [1], we impose a probability density function $f$ on $T_\mu(\mathcal{C})$ in order to avoid dealing with the nonlinearity of $\mathcal{C}$. The basic assumption here is that the support of $f$ in $T_\mu(\mathcal{C})$ is sufficiently small so that the exponential map between the support and $\mathcal{C}$ has a well-defined inverse.

2. **Finite-Dimensional Representation**: Assume that the covariance operator of the probability distribution on $T_\mu(\mathcal{C})$ has finite spectrum, and thus admits a finite representation. We approximate a tangent function by a truncated Fourier series to obtain a finite-dimensional representation. We thus characterize a probability distribution on $T_\mu(\mathcal{C})$ as that on a finite-dimensional vector space.

Let a tangent element $g \in T_\mu(\mathcal{C})$ be represented by its Fourier expansion: $g(s) = \sum_{i=1}^{m} x_i e_i(s)$, for a large positive integer $m$. Using the identification $g \equiv \mathbf{x} = \{x_i\} \in \mathbb{R}^m$, one can define a probability distribution on elements of $T_\mu(\mathcal{C})$ via a probability distribution on the coefficients $\mathbf{x}$.

We still have to decide what form does the resulting probability distribution takes. One common approach is to assume a parametric form so that learning is reduced to an estimation of the relevant parameters. As an example, a popular idea is to assume a Gaussian distribution on the underlying space. The variations of $\mathbf{x}$ as mostly restricted to an $m_1$-dimensional subspace of $\mathbb{R}^m$, called the principal subspace, for some $m_1 \le m$. On this subspace we adopt a multivariate normal with mean $\mu \in \mathbb{R}^{m_1}$ and variance $K \in \mathbb{R}^{m_1 \times m_1}$.

Estimation of $\mu$ and $K$ from the observed shapes follows the usual procedures. Computation of the mean shape $\mu$ is described in [4]. Using $\mu$ and any observed shapes $\theta_j$, we find the tangent vectors $g_j \in T_\mu(\mathcal{C})$ such that the geodesic from $\mu$ in the direction $g_j$ passes through $\theta_j$ in unit time. This tangent vector is actually computed via its finite-dimensional representation and results in the corresponding vector of coefficients $\mathbf{x}_j$. From the observed values of $\mathbf{x}_j \in \mathbb{R}^m$, one can estimate the principal subspace and the covariance matrix. Extracting the dominant eigenvectors of the estimated covariance matrix, one can capture the dominant modes of variations. The density function associated with this family of shapes is given by:

$$h(\theta; \mu, K) \equiv \frac{1}{(2\pi)^{m/2} \det(K)^{1/2}} \exp(-(\mathbf{x} - \mu)^T K^{-1} (\mathbf{x} - \mu)/2) , \qquad (3)$$

where $\Psi(\mu, g, 1) = \theta$ and $g = \sum_{i=1}^{m_1} (x_i - \mu_i) e_i(s)$.

An example of this shape learning is shown in Figure 2. The top panels show infrared pictures of tanks, followed by their extracted contours in the second row of images. These contours are then used in analyzing shapes of tanks. As an example, the 12 panels in bottom left show the observed contours of a tank when viewed from a variety of angles, and we are interested in capturing this shape variation. Repeating earlier process, the mean shape is shown in the top middle panel and the eigen values are plotted in the bottom middle panel. Twelve panels on the right show shape generated randomly from a parametric model $h(\theta; \mu, \Sigma)$.

In Figure 3 we present an interesting example of samples from three different shape models. Let the original model be $h(\theta; \mu, K)$ where $\mu$ and $K$ are as shown in Figure 2. Six samples from this model are shown in the left of Figure 3. The middle shows samples from a probability density $h(\theta; \mu, 0.2K)$ to demonstrate a smaller covariance; the samples here seem much closer to the mean shape. The right shows samples from a density where the covariance is equivariant in principal subspace, i.e. the covariance is given by $0.4\|K\|_2$ times a matrix whose top left is a $12 \times 12$ identity matrix and remaining entries are zero.

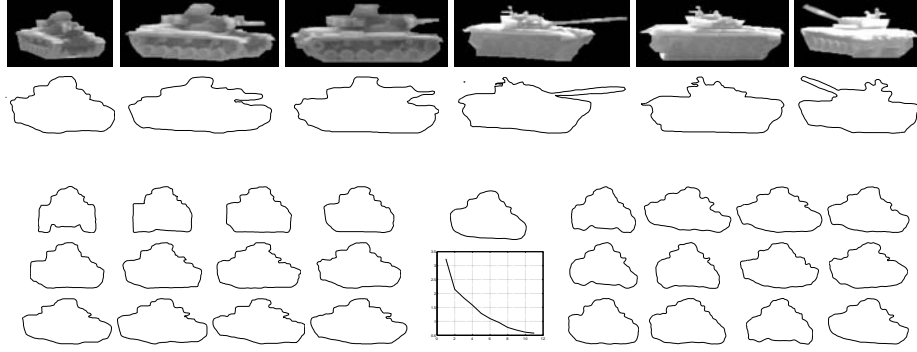

Figure 2: Top two rows: Infrared images and extracted contours of two tanks M60 and T72 at different viewing angles. Bottom row: For the 12 observed M60 shapes shown in left, the middle panels show the mean shape and the principal eigenvalues of covariance, and the right panels show 12 random samples from Gaussian model $h(\theta; \mu, K)$.

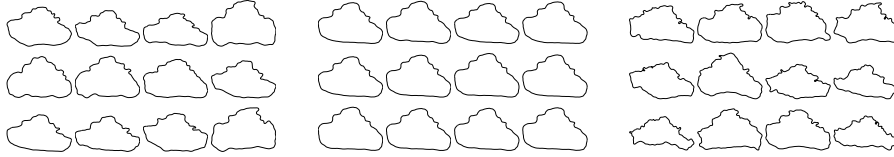

Figure 3: Comparison of samples from three families: (i) $h(\theta; \mu, K)$, (ii) $h(\theta; \mu, 0.2K)$, and (iii) $h(\theta; \mu, 0.4\|K\|_2 I_{12})$.

### 4.3 Hypothesis Testing

This framework of shape representations and statistical models on shape spaces has important applications in decision theory. One is to recognize an imaged object according to the shape of its boundary. Statistical analysis on shape spaces can be used to make a variety of decisions such as: Does this shape belong to a given family of shapes? Does these two families of shapes have similar means and variances? Given a test shape and two competing probability models, which one explains the test shape better?

We restrict to the case of binary hypothesis testing since for multiple hypotheses, one can find the best hypothesis using a sequence of binary hypothesis tests. Consider two shape families specified by their probability models: $h_1$ and $h_2$. For an observed shape $\theta \in \mathcal{C}$, we are interested in selecting one of two following hypotheses: $H_0 : \theta \sim h_1$ or $H_1 : \theta \sim h_2$. We will select a hypothesis according to the likelihood ratio test: $l(\theta) \equiv \log(\frac{h_1(\theta)}{h_2(\theta)}) \gtrless 0$. Substituting for the normal distributions (Eqn. 3) for $h_1 \equiv h(\theta; \mu_1, \Sigma_1)$ and $h_2 \equiv h(\theta; \mu_2, \Sigma_2)$, we can obtain sufficient statistics for this test. Let $\mathbf{x}_1$ be the vector of Fourier coefficients that encode the tangent direction from $\mu_1$ to $\theta$, and $\mathbf{x}_2$ be the same for direction from $\mu_2$ to $\theta$. In other words, if we let $g_1 = \sum_{i=1}^{m} x_{1,i} e_i$ and $g_2 = \sum_{i=1}^{m} x_{2,i} e_i$, then we have $\theta = \Psi(\mu_1, g_1, 1) = \Psi(\mu_2, g_2, 1)$. It follows that

$$l(\theta) = (\mathbf{x}_1^T \Sigma_1^- \mathbf{x}_1 - \mathbf{x}_2 \Sigma_2^- \mathbf{x}_2) - \frac{1}{2}(\log(\det(\Sigma_2)) - \log(\det(\Sigma_1))) \qquad (4)$$

In case the two covariances are equal to $\Sigma$, the hypothesis test reduces to $l(\theta) = (\mathbf{x}_1^T \Sigma^- \mathbf{x}_1 - \mathbf{x}_2 \Sigma^- \mathbf{x}_2) \gtrless 0$, and when $\Sigma$ is identity, the log-likelihood ratio is given by $l(\theta) = \|\mathbf{x}_1\|^2 - \|\mathbf{x}_2\|^2$. The curved nature of the shape space $\mathcal{C}$ makes the analysis

of this test difficult. For instance, one may be interested in probability of type one error but that calculation requires a probability model on $\mathbf{x}_2$ when $H_0$ is true. As a first order approximation, one can write $\mathbf{x}_2 \sim N(\bar{\mathbf{x}}, \Sigma_1)$, where $\bar{x}$ is the coefficient vector of tangent direction in $T_{\mu_2}(\mathcal{C})$ that corresponds to the geodesic from $\mu_2$ to $\mu_1$. However, the validity of this approximation remains to be tested under experimental conditions.

## 5   Conclusion

We have presented an overview of an ambitious framework for solving optimization and inference problems on a shape space. The main idea is to exploit the differential geometry of the manifold to obtain simpler solutions as compared to those obtained with PDE-based methods. We have presented some applications of this framework in image understanding. In particular, these ideas lead to a novel statistical theory of shapes of planar objects with powerful tools for shape analysis.

### Acknowledgments

This research was supported in part by grants NSF (FRG) DMS-0101429, NMA 201-01-2010, and NSF (ACT) DMS-0345242.

## References

[1] I. L. Dryden and K. V. Mardia. *Statistical Shape Analysis*. John Wiley & Son, 1998.

[2] N. Duta, M. Sonka, and A. K. Jain. Learning shape models from examples using automatic shape clustering and Procrustes analysis. In *Proceedings of Information in Medical Image Processing*, volume 1613 of *Lecture Notes in Computer Science*, pages 370–375. Springer, 1999.

[3] H. Karcher. Riemann center of mass and mollifier smoothing. *Communications on Pure and Applied Mathematics*, 30:509–541, 1977.

[4] E. Klassen, A. Srivastava, W. Mio, and S. Joshi. Analysis of planar shapes using geodesic paths on shape spaces. *IEEE Pattern Analysis and Machiner Intelligence*, 26(3):to appear, March, 2004.

[5] H. Le. Locating frechet means with application to shape spaces. *Advances in Applied Probability*, 33(2):324–338, 2001.

[6] W. Mio, A. Srivastava, and E. Klassen. Interpolation by elastica in Euclidean spaces. *Quarterly of Applied Mathematics*, to appear, 2003.

[7] D. Mumford. *Elastica and computer vision*, pages 491–506. Springer, New York, 1994.

[8] Editor: B. Romeny. *Geometry Driven Diffusions in Computer Vision*. Kluwer, 1994.

[9] T. B. Sebastian, P. N. Klein, and B. B. Kimia. On aligning curves. *IEEE Transactions on Pattern Analysis and Machine Intelligence*, 25(1):116–125, 2003.

[10] J. Sethian. *Level Set Methods: Evolving Interfaces in Geometry, Fluid Mechanics, Computer Vision, and Material Science*. Cambridge University Press, 1996.

[11] E. Sharon, A. Brandt, and R. Basri. Completion energies and scale. *IEEE Transactions on Pattern Analysis and Machine Intelligence*, 22(10):1117–1131, 2000.

[12] L. Younes. Optimal matching between shapes via elastic deformations. *Journal of Image and Vision Computing*, 17(5/6):381–389, 1999.
